# Generalizing from Several Related Classification Tasks to a New Unlabeled Sample

**Gilles Blanchard**
Universität Potsdam
blanchard@math.uni-potsdam.de

**Gyemin Lee, Clayton Scott**
University of Michigan
{gyemin,clayscot}@umich.edu

## Abstract

We consider the problem of assigning class labels to an unlabeled test data set, given several labeled training data sets drawn from similar distributions. This problem arises in several applications where data distributions fluctuate because of biological, technical, or other sources of variation. We develop a distribution-free, kernel-based approach to the problem. This approach involves identifying an appropriate reproducing kernel Hilbert space and optimizing a regularized empirical risk over the space. We present generalization error analysis, describe universal kernels, and establish universal consistency of the proposed methodology. Experimental results on flow cytometry data are presented.

## 1 Introduction

Is it possible to leverage the solution of one classification problem to solve another? This is a question that has received increasing attention in recent years from the machine learning community, and has been studied in a variety of settings, including multi-task learning, covariate shift, and transfer learning. In this work we study a new setting for this question, one that incorporates elements of the three aforementioned settings, and is motivated by many practical applications.

To state the problem, let $\mathcal{X}$ be a feature space and $\mathcal{Y}$ a space of labels to predict; to simplify the exposition, we will assume the setting of binary classification, $\mathcal{Y} = \{-1, 1\}$, although the methodology and results presented here are valid for general output spaces. For a given distribution $P_{XY}$, we refer to the $X$ marginal distribution $P_X$ as simply the marginal distribution, and the conditional $P_{XY}(Y|X)$ as the posterior distribution.

There are $N$ similar but distinct distributions $P_{XY}^{(i)}$ on $\mathcal{X} \times \mathcal{Y}$, $i = 1, \ldots, N$. For each $i$, there is a training sample $S_i = (X_{ij}, Y_{ij})_{1 \le j \le n_i}$ of iid realizations of $P_{XY}^{(i)}$. There is also a test distribution $P_{XY}^T$ that is similar to but again distinct from the "training distributions" $P_{XY}^{(i)}$. Finally, there is a test sample $(X_j^T, Y_j^T)_{1 \le j \le n_T}$ of iid realizations of $P_{XY}^T$, but in this case the labels $Y_j$ are not observed.

The goal is to correctly predict these unobserved labels. Essentially, given a random sample from the marginal test distribution $P_X^T$, we would like to predict the corresponding labels. Thus, when we say that the training and test distributions are "similar," we mean that there is some pattern making it possible to learn a mapping from marginal distributions to labels. We will refer to this learning problem as *learning marginal predictors*. A concrete motivating application is given below.

This problem may be contrasted with other learning problems. In multi-task learning, only the training distributions are of interest, and the goal is to use the similarity among distributions to improve the training of individual classifiers [1, 2, 3]. In our context, we view these distributions as "training tasks," and seek to generalize to a new distribution/task. In the covariate shift problem, the marginal test distribution is different from the marginal training distribution(s), but the posterior

distribution is assumed to be the same [4]. In our case, both marginal and posterior test distributions can differ from their training counterparts [5].

Finally, in transfer learning, it is typically assumed that at least a few labels are available for the test data, and the training data sets are used to improve the performance of a standard classifier, for example by learning a metric or embedding which is appropriate for all data sets [6, 7]. In our case, no test labels are available, but we hope that through access to multiple training data sets, it is still possible to obtain collective knowledge about the "labeling process" that may be transferred to the test distribution. Some authors have considered transductive transfer learning, which is similar to the problem studied here in that no test labels are available. However, existing work has focused on the case $N = 1$ and typically relies on the covariate shift assumption [8].

We propose a distribution-free, kernel-based approach to the problem of learning marginal predictors. Our methodology is shown to yield a consistent learning procedure, meaning that the generalization error tends to the best possible as the sample sizes $N, \{n_i\}, n_T$ tend to infinity. We also offer a proof-of-concept experimental study validating the proposed approach on flow cytometry data, including comparisons to multi-task kernels and a simple pooling approach.

## 2   Motivating Application: Automatic Gating of Flow Cytometry Data

Flow cytometry is a high-throughput measurement platform that is an important clinical tool for the diagnosis of many blood-related pathologies. This technology allows for quantitative analysis of individual cells from a given population, derived for example from a blood sample from a patient. We may think of a flow cytometry data set as a set of $d$-dimensional attribute vectors $(X_j)_{1 \leq j \leq n}$, where $n$ is the number of cells analyzed, and $d$ is the number of attributes recorded per cell. These attributes pertain to various physical and chemical properties of the cell. Thus, a flow cytometry data set is a random sample from a patient-specific distribution.

Now suppose a pathologist needs to analyze a new ("test") patient with data $(X_j^T)_{1 \leq j \leq n_T}$. Before proceeding, the pathologist first needs the data set to be "purified" so that only cells of a certain type are present. For example, lymphocytes are known to be relevant for the diagnosis of leukemia, whereas non-lymphocytes may potentially confound the analysis. In other words, it is necessary to determine the label $Y_j^T \in \{-1, 1\}$ associated to each cell, where $Y_j^T = 1$ indicates that the $j$-th cell is of the desired type.

In clinical practice this is accomplished through a manual process known as "gating." The data are visualized through a sequence of two-dimensional scatter plots, where at each stage a line segment or polygon is manually drawn to eliminate a portion of the unwanted cells. Because of the variability in flow cytometry data, this process is difficult to quantify in terms of a small subset of simple rules. Instead, it requires domain-specific knowledge and iterative refinement. Modern clinical laboratories routinely see dozens of cases per day, so it would be desirable to automate this process.

Since clinical laboratories maintain historical databases, we can assume access to a number ($N$) of historical patients that have already been expert-gated. Because of biological and technical variations in flow cytometry data, the distributions $P_{XY}^{(i)}$ of the historical patients will vary. For example, Fig. 1 shows exemplary two-dimensional scatter plots for two different patients, where the shaded cells correspond to lymphocytes. Nonetheless, there are certain general trends that are known to hold for all flow cytometry measurements. For example, lymphocytes are known to exhibit low levels of the "side-scatter" (SS) attribute, while expressing high levels of the attribute CD45 (see column 2 of Fig. 1). More generally, virtually every cell type of interest has a known tendency (e.g., high or low) for most measured attributes. Therefore, it is reasonable to assume that there is an underlying distribution (on distributions) governing flow cytometry data sets, that produces roughly similar distributions thereby making possible the automation of the gating process.

## 3   Formal Setting

Let $\mathcal{X}$ denote the observation space and $\mathcal{Y} = \{-1, 1\}$ the output space. Let $\mathfrak{P}_{\mathcal{X} \times \mathcal{Y}}$ denote the set of probability distributions on $\mathcal{X} \times \mathcal{Y}$, $\mathfrak{P}_{\mathcal{X}}$ the set of probability distributions on $\mathcal{X}$, and $\mathfrak{P}_{\mathcal{Y}|\mathcal{X}}$ the set of conditional probabilities of $Y$ given $X$ (also known as Markov transition kernels from $X$ to

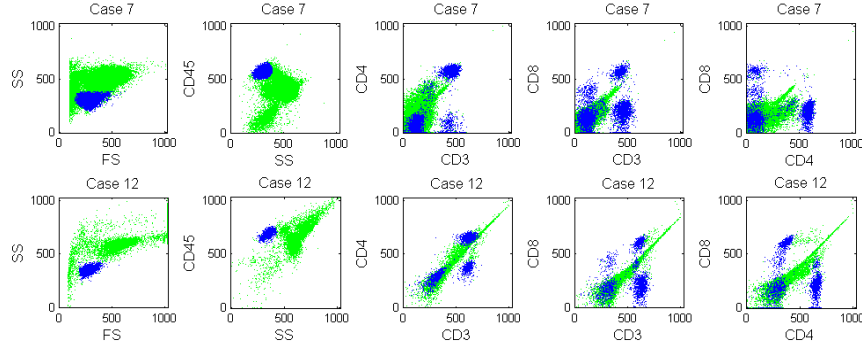

Figure 1: Two-dimensional projections of multi-dimensional flow cytometry data. Each row corresponds to a single patient. The distribution of cells differs from patient to patient. Lymphocytes, a type of white blood cell, are marked dark (blue) and others are marked bright (green). These were manually selected by a domain expert.

$Y$) which we also call "posteriors" in this work. The disintegration theorem (see for instance [9], Theorem 6.4) tells us that (under suitable regularity properties, e.g., $\mathcal{X}$ is a Polish space) any element $P_{XY} \in \mathfrak{P}_{\mathcal{X} \times \mathcal{Y}}$ can be written as a product $P_{XY} = P_X \bullet P_{Y|X}$, with $P_X \in \mathfrak{P}_{\mathcal{X}}$, $P_{Y|X} \in \mathfrak{P}_{Y|X}$. The space $\mathfrak{P}_{\mathcal{X} \times \mathcal{Y}}$ is endowed with the topology of weak convergence and the associated Borel $\sigma$-algebra.

It is assumed that there exists a distribution $\mu$ on $\mathfrak{P}_{\mathcal{X} \times \mathcal{Y}}$, where $P_{XY}^{(1)}, \ldots, P_{XY}^{(N)}$ are i.i.d. realizations from $\mu$, and the sample $S_i$ is made of $n_i$ i.i.d. realizations of $(X, Y)$ following the distribution $P_{XY}^{(i)}$. Now consider a test distribution $P_{XY}^T$ and test sample $S^T = (X_j^T, Y_j^T)_{1 \leq j \leq n_T}$, whose labels are not observed. A decision function is a function $f : \mathfrak{P}_{\mathcal{X}} \times \mathcal{X} \mapsto \mathbb{R}$ that predicts $\widehat{Y}_i = f(\widehat{P}_X, X_i)$, where $\widehat{P}_X$ is the associated empirical $X$ distribution. If $\ell : \mathbb{R} \times \mathcal{Y} \mapsto \mathbb{R}_+$ is a loss, then the average loss incurred on the test sample is $\frac{1}{n_T} \sum_{i=1}^{n_T} \ell(\widehat{Y}_i^T, Y_i^T)$. Based on this, we define the average generalization error of a decision function over test samples of size $n_T$,

$$\mathcal{E}(f, n_T) := \mathbb{E}_{P_{XY}^T \sim \mu} \mathbb{E}_{S^T \sim (P_{XY}^T)^{\otimes n_T}} \left[ \frac{1}{n_T} \sum_{i=1}^{n_T} \ell(f(\widehat{P}_X^T, X_i^T), Y_i^T) \right]. \tag{1}$$

In important point of the analysis is that, at training time as well as at test time, the marginal distribution $P_X$ for a sample is only known through the sample itself, that is, through the empirical marginal $\widehat{P}_X$. As is clear from equation (1), because of this the generalization error also depends on the test sample size $n_T$. As $n_T$ grows, $\widehat{P}_X^T$ will converge to $P_X^T$. This motivates the following generalization error when we have an infinite test sample, where we then assume that the true marginal $P_X^T$ is observed:

$$\mathcal{E}(f, \infty) := \mathbb{E}_{P_{XY}^T \sim \mu} \mathbb{E}_{(X^T, Y^T) \sim P_{XY}^T} \left[ \ell(f(P_X^T, X^T), Y^T) \right]. \tag{2}$$

To gain some insight into this risk, let us decompose $\mu$ into two parts, $\mu_X$ which generates the marginal distribution $P_X$, and $\mu_{Y|X}$ which, conditioned on $P_X$, generates the posterior $P_{Y|X}$. Denote $\tilde{X} = (P_X, X)$. We then have

$$\mathcal{E}(f, \infty) = \mathbb{E}_{P_X \sim \mu_X} \mathbb{E}_{P_{Y|X} \sim \mu_{Y|X}} \mathbb{E}_{X \sim P_X} \mathbb{E}_{Y|X \sim P_{Y|X}} \left[ \ell(f(\tilde{X}), Y) \right]$$

$$= \mathbb{E}_{P_X \sim \mu_X} \mathbb{E}_{X \sim P_X} \mathbb{E}_{P_{Y|X} \sim \mu_{Y|X}} \mathbb{E}_{Y|X \sim P_{Y|X}} \left[ \ell(f(\tilde{X}), Y) \right]$$

$$= \mathbb{E}_{(\tilde{X}, Y) \sim Q^\mu} \left[ \ell(f(\tilde{X}), Y) \right].$$

Here $Q^\mu$ is the distribution that generates $\tilde{X}$ by first drawing $P_X$ according to $\mu_X$, and then drawing $X$ according to $P_X$. Similarly, $Y$ is generated, conditioned on $\tilde{X}$, by first drawing $P_{Y|X}$ according to $\mu_{Y|X}$, and then drawing $Y$ from $P_{Y|X}$. From this last expression, we see that the risk is like a standard binary classification risk based on $(\tilde{X}, Y) \sim Q^\mu$. Thus, we can deduce several properties

that are known to hold for binary classification risks. For example, if the loss is the 0/1 loss, then $f^*(\tilde{X}) = 2\tilde{\eta}(\tilde{X}) - 1$ is an optimal predictor, where $\tilde{\eta}(\tilde{X}) = \mathbb{E}_{Y \sim Q^\mu_{Y|\tilde{X}}}\left[\mathbf{1}_{\{Y=1\}}\right]$. More generally,

$$\mathcal{E}(f, \infty) - \mathcal{E}(f^*, \infty) = \mathbb{E}_{\tilde{X} \sim Q^\mu_{\tilde{X}}}\left[\mathbf{1}_{\{\text{sign}(f(\tilde{X})) \neq \text{sign}(f^*(\tilde{X}))\}} |2\tilde{\eta}(\tilde{X}) - 1|\right].$$

Our goal is a learning rule that asymptotically predicts as well as the global minimizer of (2), for a *general* loss $\ell$. By the above observations, consistency with respect to a general $\ell$ (thought of as a surrogate) will imply consistency for the 0/1 loss, provided $\ell$ is classification calibrated [10]. Despite the similarity to standard binary classification in the infinite sample case, we emphasize that the learning task here is different, because the realizations $(\tilde{X}_{ij}, Y_{ij})$ are neither independent nor identically distributed.

Finally, we note that there is a condition where for $\mu$-almost all test distribution $P^T_{XY}$, the classifier $f^*(P^T_X, .)$ (where $f^*$ is the global minimizer of (2)) coincides with the optimal Bayes classifier for $P^T_{XY}$, although *no labels from this test distribution are observed*. This condition is simply that the posterior $P_{Y|X}$ is ($\mu$-almost surely) a function of $P_X$. In other words, with the notation introduced above, $\mu_{Y|X}(P_X)$ is a Dirac delta for $\mu$-almost all $P_X$. Although we will *not* be assuming this condition throughout the paper, it is implicitly assumed in the motivating application presented in Section 2, where an expert labels the data points by just looking at their marginal distribution.

**Lemma 3.1.** *For a fixed distribution $P_{XY}$, and a decision function $f : \mathcal{X} \to \mathbb{R}$, let us denote $\mathcal{R}(f, P_{XY}) = \mathbb{E}_{(X,Y) \sim P_{XY}}\left[\ell(f(X), Y)\right]$ and*

$$\mathcal{R}^*(P_{XY}) := \min_{f:\mathcal{X} \to \mathbb{R}} \mathcal{R}(f, P_{XY}) = \min_{f:\mathcal{X} \to \mathbb{R}} \mathbb{E}_{(X,Y) \sim P_{XY}}\left[\ell(f(X), Y)\right]$$

*the corresponding optimal (Bayes) risk for the loss function $\ell$. Assume that $\mu$ is a distribution on $\mathfrak{P}_{\mathcal{X} \times \mathcal{Y}}$ such that $\mu$-a.s. it holds $P_{Y|X} = F(P_X)$ for some deterministic mapping $F$. Let $f^*$ be a minimizer of the risk (2). Then we have for $\mu$-almost all $P_{XY}$:*

$$\mathcal{R}(f^*(P_X, .), P_{XY}) = \mathcal{R}^*(P_{XY})$$

*and*

$$\mathcal{E}(f^*, \infty) = \mathbb{E}_{P_{XY} \sim \mu}\left[\mathcal{R}^*(P_{XY})\right].$$

*Proof.* Straightforward. Obviously for any $f : \mathfrak{P}_{\mathcal{X}} \times \mathcal{X} \to \mathbb{R}$, one has for all $P_{XY}$: $\mathcal{R}(f(P_X, .), P_{XY}) \geq \mathcal{R}^*(P_{XY})$. For any fixed $P_X \in \mathfrak{P}_{\mathcal{X}}$, consider $P_{XY} := P_X \bullet F(P_X)$ and $g^*(P_X)$ a Bayes classifier for this joint distribution. Pose $f(P_X, x) := g^*(P_X)(x)$. Then $f$ coincides for $\mu$-almost $P_{XY}$ with a Bayes classifier for $P_{XY}$, achieving equality in the above inequality. The second equality follows by taking expectation over $P_{XY} \sim \mu$. □

## 4 Learning Algorithm

We consider an approach based on positive semi-definite kernels, or simply kernels for short. Background information on kernels, including the definition, normalized kernels, universal kernels, and reproducing kernel Hilbert spaces (RKHSs), may be found in [11]. Several well-known learning algorithms, such as support vector machines and kernel ridge regression, may be viewed as minimizers of a norm-regularized empirical risk over the RKHS of a kernel. A similar development also exists for multi-task learning [3]. Inspired by this framework, we consider a general kernel algorithm as follows.

Consider the loss function $\ell : \mathbb{R} \times \mathcal{Y} \to \mathbb{R}_+$. Let $\overline{k}$ be a kernel on $\mathfrak{P}_X \times \mathcal{X}$, and let $\mathcal{H}_{\overline{k}}$ be the associated RKHS. For the sample $S_i$ let $\widehat{P}^{(i)}_X$ denote the corresponding empirical distribution of the $X_{ij}$s. Also consider the extended input space $\mathfrak{P}_X \times \mathcal{X}$ and the extended data $\tilde{X}_{ij} = (\widehat{P}^{(i)}_X, X_{ij})$. Note that $\widehat{P}^{(i)}_X$ plays a role similar to the task index in multi-task learning. Now define

$$\widehat{f}_\lambda = \arg\min_{f \in \mathcal{H}_{\overline{k}}} \frac{1}{N} \sum_{i=1}^N \frac{1}{n_i} \sum_{j=1}^{n_i} \ell(f(\tilde{X}_{ij}), Y_{ij}) + \lambda \|f\|^2. \tag{3}$$

For the hinge loss, by the representer theorem [12] this optimization problem reduces to a quadratic program equivalent to the dual of a kind of cost-sensitive SVM, and therefore can be solved using existing software packages. The final predictor has the form

$$\widehat{f}_\lambda(\widehat{P}_X, x) = \sum_{i=1}^{N} \sum_{j=1}^{n_i} \alpha_{ij} Y_{ij} \overline{k}((\widehat{P}_X^{(i)}, X_{ij}), (\widehat{P}_X, x))$$

where the $\alpha_{ij}$ are nonnegative and mostly zero. See [11] for details.

In the rest of the paper we will consider a kernel $\overline{k}$ on $\mathfrak{P}_\mathcal{X} \times \mathcal{X}$ of the product form

$$\overline{k}((P_1, x_1), (P_2, x_2)) = k_P(P_1, P_2) k_X(x_1, x_2), \tag{4}$$

where $k_P$ is a kernel on $\mathfrak{P}_\mathcal{X}$ and $k_X$ a kernel on $\mathcal{X}$. Furthermore, we will consider kernels on $\mathfrak{P}_\mathcal{X}$ of a particular form. Let $k'_X$ denote a kernel on $\mathcal{X}$ (which might be different from $k_X$) that is measurable and bounded. We define the following mapping $\Psi : \mathfrak{P}_\mathcal{X} \to \mathcal{H}_{k'_X}$:

$$P_X \mapsto \Psi(P_X) := \int_\mathcal{X} k'_X(x, \cdot) dP_X(x). \tag{5}$$

This mapping has been studied in the framework of "characteristic kernels" [13], and it has been proved that there are important links between universality of $k'_X$ and injectivity of $\Psi$ [14, 15].

Note that the mapping $\Psi$ is linear. Therefore, if we consider the kernel $k_P(P_X, P'_X) = \langle \Psi(P_X), \Psi(P'_X) \rangle$, it is a linear kernel on $\mathfrak{P}_\mathcal{X}$ and cannot be a universal kernel. For this reason, we introduce yet another kernel $\mathfrak{K}$ on $\mathcal{H}_{k'_X}$ and consider the kernel on $\mathfrak{P}_\mathcal{X}$ given by

$$k_P(P_X, P'_X) = \mathfrak{K}\left(\Psi(P_X), \Psi(P'_X)\right). \tag{6}$$

Note that particular kernels inspired by the finite dimensional case are of the form

$$\mathfrak{K}(v, v') = F(\|v - v'\|), \tag{7}$$

or

$$\mathfrak{K}(v, v') = G(\langle v, v' \rangle), \tag{8}$$

where $F, G$ are real functions of a real variable such that they define a kernel. For example, $F(t) = \exp(-t^2/(2\sigma^2))$ yields a Gaussian-like kernel, while $G(t) = (1 + t)^d$ yields a polynomial-like kernel. Kernels of the above form on the space of probability distributions over a compact space $\mathcal{X}$ have been introduced and studied in [16]. Below we apply their results to deduce that $\overline{k}$ is a universal kernel for certain choices of $k_X, k'_X$, and $\mathfrak{K}$.

## 5   Learning Theoretic Study

Although the regularized estimation formula (3) defining $\widehat{f}_\lambda$ is standard, the generalization error analysis is not, since the $\widetilde{X}_{ij}$ are neither identically distributed nor independent. We begin with a generalization error bound that establishes uniform estimation error control over functions belonging to a ball of $\mathcal{H}_{\overline{k}}$. We then discuss universal kernels, and finally deduce universal consistency of the algorithm. To simplify somewhat the analysis, we assume below that all training samples have the same size $n_i = n$. Also let $\mathcal{B}_k(r)$ denote the closed ball of radius $r$, centered at the origin, in the RKHS of the kernel $k$. We consider the following assumptions on the loss and kernels:

**(Loss)** The loss function $\ell : \mathbb{R} \times \mathcal{Y} \to \mathbb{R}_+$ is $L_\ell$-Lipschitz in its first variable and bounded by $B_\ell$.

**(Kernels-A)** The kernels $k_X, k'_X$ and $\mathfrak{K}$ are bounded respectively by constants $B_k^2, B_{k'}^2 \geq 1$, and $B_{\mathfrak{K}}^2$. In addition, the canonical feature map $\Phi_{\mathfrak{K}} : \mathcal{H}_{k'_X} \to \mathcal{H}_{\mathfrak{K}}$ associated to $\mathfrak{K}$ satisfies a Hölder condition of order $\alpha \in (0, 1]$ with constant $L_{\mathfrak{K}}$, on $\mathcal{B}_{k'_X}(B_{k'})$:

$$\forall v, w \in \mathcal{B}_{k'_X}(B_{k'}): \qquad \|\Phi_{\mathfrak{K}}(v) - \Phi_{\mathfrak{K}}(w)\| \leq L_{\mathfrak{K}} \|v - w\|^\alpha. \tag{9}$$

Sufficient conditions for (9) are described in [11]. As an example, the condition is shown to hold with $\alpha = 1$ when $\mathfrak{K}$ is the Gaussian-like kernel on $\mathcal{H}_{k'_X}$. The boundedness assumptions are also clearly satisfied for Gaussian kernels.

**Theorem 5.1** (Uniform estimation error control). *Assume conditions* **(Loss)** *and* **(Kernels-A)** *hold. If $P_{XY}^{(1)}, \ldots, P_{XY}^{(N)}$ are i.i.d. realizations from $\mu$, and for each $i = 1, \ldots, N$, the sample $S_i = (X_{ij}, Y_{ij})_{1 \le j \le n}$ is made of i.i.d. realizations from $P_{XY}^{(i)}$, then for any $R > 0$, with probability at least $1 - \delta$:*

$$\sup_{f \in \mathcal{B}_{\overline{k}}(R)} \left| \frac{1}{N} \sum_{i=1}^{N} \frac{1}{n} \sum_{j=1}^{n} \ell(f(\widetilde{X}_{ij}), Y_{ij}) - \mathcal{E}(f, \infty) \right|$$

$$\le c \left( R B_k L_\ell \left( B_{k'} L_{\mathfrak{K}} \left( \frac{\log N + \log \delta^{-1}}{n} \right)^{\frac{\alpha}{2}} + B_{\mathfrak{K}} \frac{1}{\sqrt{N}} \right) + B_\ell \sqrt{\frac{\log \delta^{-1}}{N}} \right), \quad (10)$$

*where $c$ is a numerical constant, and $\mathcal{B}_{\overline{k}}(R)$ denotes the ball of radius $R$ of $\mathcal{H}_{\overline{k}}$.*

*Proof sketch.* The full proofs of this and other results are given in [11]. We give here a brief overview. We use the decomposition

$$\sup_{f \in \mathcal{B}_{\overline{k}}(R)} \left| \frac{1}{N} \sum_{i=1}^{N} \frac{1}{n_i} \sum_{j=1}^{n_i} \ell(f(\widetilde{X}_{ij}), Y_{ij}) - \mathcal{E}(f, \infty) \right|$$

$$\le \sup_{f \in \mathcal{B}_{\overline{k}}(R)} \left| \frac{1}{N} \sum_{i=1}^{N} \frac{1}{n_i} \sum_{j=1}^{n_i} \left( \ell(f(\widehat{P}_X^{(i)}, X_{ij}), Y_{ij}) - \ell(f(P_X^{(i)}, X_{ij}), Y_{ij}) \right) \right|$$

$$+ \sup_{f \in \mathcal{B}_{\overline{k}}(R)} \left| \frac{1}{N} \sum_{i=1}^{N} \frac{1}{n_i} \sum_{j=1}^{n_i} \ell(f(P_X^{(i)}, X_{ij}), Y_{ij}) - \mathcal{E}(f, \infty) \right| =: (I) + (II).$$

Bounding (I), using the Lipschitz property of the loss function, can be reduced to controlling

$$\left\| f(\widehat{P}_X^{(i)}, .) - f(P_X^{(i)}, .) \right\|_\infty,$$

conditional to $P_X^{(i)}$, uniformly for $i = 1, \ldots, N$. This can be obtained using the reproducing property of the kernel $\overline{k}$, the convergence of $\Psi(\widehat{P}_X^{(i)})$ to $\Psi(P_X^{(i)})$ as a consequence of Hoeffding's inequality in a Hilbert space, and the other assumptions (boundedness/Hölder property) on the kernels.

Concerning the control of the term (II), it can be decomposed in turn into the convergence conditional to $(P_X^{(i)})$, and the convergence of the conditional generalization error. In both cases, a standard approach using the Azuma-McDiarmid's inequality [17] followed by symmetrization and Rademacher complexity analysis on a kernel space [18, 19] can be applied. For the first part, the random variables are the $(X_{ij}, Y_{ij})$ (which are independent conditional to $(P_X^{(i)})$); for the second part, the i.i.d. variables are the $(P_X^{(i)})$ (the $(X_{ij}, Y_{ij})$ being integrated out). $\qquad \square$

To establish that $\overline{k}$ is universal on $\mathfrak{P}_{\mathcal{X}} \times \mathcal{X}$, the following lemma is useful.

**Lemma 5.2.** *Let $\Omega, \Omega'$ be two compact spaces and $k, k'$ be kernels on $\Omega, \Omega'$, respectively. If $k, k'$ are both universal, then the product kernel*

$$\overline{k}((x, x'), (y, y')) := k(x, y)k'(x', y')$$

*is universal on $\Omega \times \Omega'$.*

Several examples of universal kernels are known on Euclidean space. We also need universal kernels on $\mathfrak{P}_{\mathcal{X}}$. Fortunately, this was recently investigated [16]. Some additional assumptions on the kernels and feature space are required:

**(Kernels-B)** $k_X, k'_X, \mathfrak{K}$, and $\mathcal{X}$ satisfy the following: $\mathcal{X}$ is a compact metric space; $k_X$ is universal on $\mathcal{X}$; $k'_X$ is continuous and universal on $\mathcal{X}$; $\mathfrak{K}$ is universal on any compact subset of $\mathcal{H}_{k'_X}$.

Adapting the results of [16], we have the following.

**Theorem 5.3** (Universal kernel). *Assume condition* (**Kernels-B**) *holds. Then, for $k_P$ defined as in* (6)*, the product kernel $\overline{k}$ in* (4) *is universal on $\mathfrak{P}_{\mathcal{X}} \times \mathcal{X}$. Furthermore, the assumption on $\mathfrak{K}$ is fulfilled if $\mathfrak{K}$ is of the form* (8)*, where $G$ is an analytical function with positive Taylor series coefficients, or if $\mathfrak{K}$ is the normalized kernel associated to such a kernel.*

As an example, suppose that $\mathcal{X}$ is a compact subset of $\mathbb{R}^d$. Let $k_X$ and $k'_X$ be Gaussian kernels on $\mathcal{X}$. Taking $G(t) = \exp(t)$, it follows that $\mathfrak{K}(P_X, P'_X) = \exp(\langle \Psi(P_X), \Psi(P'_X) \rangle_{\mathcal{H}_{k'_X}})$ is universal on $\mathfrak{P}_{\mathcal{X}}$. By similar reasoning as in the finite dimensional case, the Gaussian-like kernel $\mathfrak{K}(P_X, P'_X) = \exp(-\frac{1}{2\sigma^2}\|\Psi(P_X) - \Psi(P'_X)\|^2_{\mathcal{H}_{k'_X}})$ is also universal on $\mathfrak{P}_{\mathcal{X}}$. Thus the product kernel is universal.

**Corollary 5.4** (Universal consistency). *Assume the conditions* (**Loss**)*,* (**Kernels-A**)*, and* (**Kernels-B**) *are satisfied. Assume that $N, n$ grow to infinity in such a way that $N = \mathcal{O}(n^\gamma)$ for some $\gamma > 0$. Then, if $\lambda_j$ is a sequence such that $\lambda_j \to 0$ and $\lambda_j \sqrt{\frac{j}{\log j}} \to \infty$, it holds that*

$$\mathcal{E}(\widehat{f}_{\lambda_{\min(N,n^\alpha)}}, \infty) \to \inf_{f:\mathfrak{P}_{\mathcal{X}} \times \mathcal{X} \to \mathbb{R}} \mathcal{E}(f, \infty)$$

*in probability.*

## 6 Experiments

We demonstrate the proposed methodology for flow cytometry data auto-gating, described above. Peripheral blood samples were obtained from 35 normal patients, and lymphocytes were classified by a domain expert. The corresponding flow cytometry data sets have sample sizes ranging from 10,000 to 100,000, and the proportion of lymphocytes in each data set ranges from 10 to 40%. We took $N = 10$ of these data sets for training, and the remaining 25 for testing. To speed training time, we subsampled the 10 training data sets to have 1000 data points (cells) each. Adopting the hinge loss, we used the SVM$^{light}$ [20] package to solve the quadratic program characterizing the solution.

The kernels $k_X$, $k'_X$, and $\mathfrak{K}$ are all taken to be Gaussian kernels with respective bandwidths $\sigma_X$, $\sigma'_X$, and $\sigma$. We set $\sigma_X$ such that $\sigma_X^2$ equals 10 times the average distance of a data point to its nearest neighbor within the same data set. The second bandwidth was defined similarly, while the third was set to 1. The regularization parameter $\lambda$ was set to 1.

| $k_P$ | Train | Test |
|---|---|---|
| Pooling ($\tau = 1$) | 1.41 | 2.32 |
| MTL ($\tau = 0.01$) | 1.59 | 2.64 |
| MTL ($\tau = 0.5$) | 1.34 | 2.36 |
| Proposed | 1.32 | 2.29 |

Table 1: The misclassification rates (%) on training data sets and test data sets for different $k_P$. The proposed method adapts the decision function to the test data (through the marginal-dependent kernel), accounting for its improved performance.

For comparison, we also considered three other options for $k_P$. These kernels have the form $k_P(P_1, P_2) = 1$ if $P_1 = P_2$, and $k_P(P_1, P_2) = \tau$ otherwise. When $\tau = 1$, the method is equivalent to pooling all of the training data together in one data set, and learning a single SVM classifier. This idea has been previously studied in the context of flow cytometry by [21]. When $0 < \tau < 1$, we obtain a kernel like what was used for multi-task learning (MTL) by [3]. Note that these kernels have the property that if $P_1$ is a training data set, and $P_2$ a test data set, then $P_1 \neq P_2$ and so $k_P(P_1, P_2)$ is simply a constant. This implies that the learning rules produced by these kernels do not adapt to the test distribution, unlike the proposed kernel. In the experiments, we take $\tau = 1$ (pooling), 0.01, and 0.5 (MTL).

The results are shown in Fig. 2 and summarized in Table 1. The middle column of the table reports the average misclassification rate on the training data sets. Here we used those data points that were not part of the 1000-element subsample used for training. The right column shows the average misclassification rate on the test data sets.

## 7 Discussion

Our approach to learning marginal predictors relies on the extended input pattern $\tilde{X} = (P_X, X)$. Thus, we study the natural algorithm of minimizing a regularized empirical loss over a reproducing

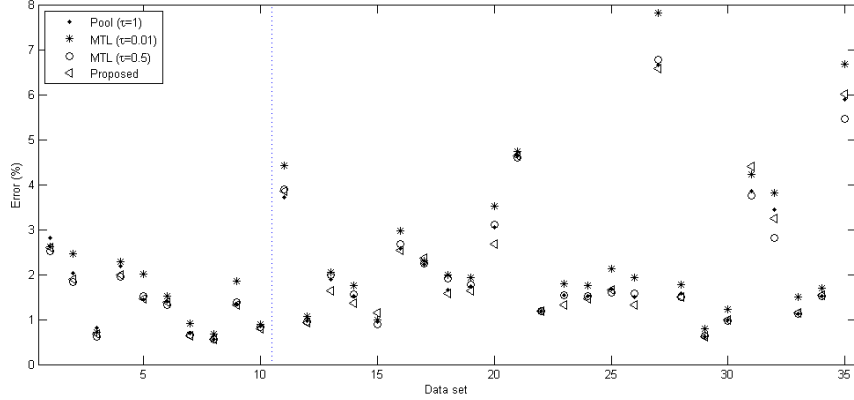

Figure 2: The misclassification rates (%) on training data sets and test data sets for different $k_P$. The last 25 data sets separated by dotted line are not used during training.

kernel Hilbert space associated with the extended input domain $\mathfrak{P}_\mathcal{X} \times \mathcal{X}$. We also establish universal consistency, using a novel generalization error analysis under the inherent non-iid sampling plan, and a construction of a universal kernel on $\mathfrak{P}_\mathcal{X} \times \mathcal{X}$. For the hinge loss, the algorithm may be implemented using standard techniques for SVMs. The algorithm is applied to flow cytometry auto-gating, and shown to improve upon kernels that do not adapt to the test distribution.

Several future directions exist. From an application perspective, the need for adaptive classifiers arises in many applications, especially in biomedical applications involving biological and/or technical variation in patient data. For example, when electrocardiograms are used to monitor cardiac patients, it is desirable to classify each heartbeat as irregular or not. However, irregularities in a test patient's heartbeat will differ from irregularities of historical patients, hence the need to adapt to the test distribution [22].

We can also ask how the methodology and analysis can be extended to the context where a small number of labels are available for the test distribution, as is commonly assumed in transfer learning. In this setting, two approaches are possible. The simplest one is to use the same optimization problem (3), wherein we include additionally the labeled examples of the test distribution. However, if several test samples are to be treated in succession, and we want to avoid a full, resource-consuming re-training using all the training samples each time, an interesting alternative is the following: learn once a function $f_0(P_X, x)$ using the available training samples via (3); then, given a partially labeled test sample, learn a decision function on this sample only via the usual kernel norm regularized empirical loss minimization method, but replace the usual regularizer term $\|f\|^2$ by $\|f - f_0(P_x, .)\|^2$ (note that $f_0(P_x, .) \in \mathcal{H}_{\overline{k}}$). In this sense, the marginal-adaptive decision function learned from the training samples would serve as a "prior" for learning on the test data.

It would also be of interest to extend the proposed methodology to a multi-class setting. In this case, the problem has an interesting interpretation in terms of "learning to cluster." Each training task may be viewed as a data set that has been clustered by a teacher. Generalization then entails the ability to learn the clustering process, so that clusters may be assigned to a new unlabeled data set.

Future work may consider other asymptotic regimes, e.g., where $\{n_i\}, n_T$ do not tend to infinity, or they tend to infinity much slower than $N$. It may also be of interest to develop implementations for differentiable losses such as the logistic loss, allowing for estimation of posterior probabilities. Finally, we would like to specify conditions on $\mu$, the distribution-generating distribution, that are favorable for generalization (beyond the simple condition discussed in Lemma 3.1).

**Acknowledgments**

G. Blanchard was supported by the European Community's 7th Framework Programme under the PASCAL2 Network of Excellence (ICT-216886) and under the E.U. grant agreement 247022 (MASH Project). G. Lee and C. Scott were supported in part by NSF Grant No. 0953135.

# References

[1] S. Thrun, "Is learning the n-th thing any easier than learning the first?," *Advances in Neural Information Processing Systems*, pp. 640–646, 1996.

[2] R. Caruana, "Multitask learning," *Machine Learning*, vol. 28, pp. 41–75, 1997.

[3] T. Evgeniou and M. Pontil, "Learning multiple tasks with kernel methods," *J. Machine Learning Research*, pp. 615–637, 2005.

[4] S. Bickel, M. Brückner, and T. Scheffer, "Discriminative learning under covariate shift," *J. Machine Learning Research*, pp. 2137–2155, 2009.

[5] J. Quionero-Candela, M. Sugiyama, A. Schwaighofer, and N. D. Lawrence, *Dataset Shift in Machine Learning*, The MIT Press, 2009.

[6] R. K. Ando and T. Zhang, "A high-performance semi-supervised learning method for text chunking," *Proceedings of the 43rd Annual Meeting on Association for Computational Linguistics (ACL 05)*, pp. 1–9, 2005.

[7] A. Rettinger, M. Zinkevich, and M. Bowling, "Boosting expert ensembles for rapid concept recall," *Proceedings of the 21st National Conference on Artificial Intelligence (AAAI 06)*, vol. 1, pp. 464–469, 2006.

[8] A. Arnold, R. Nallapati, and W.W. Cohen, "A comparative study of methods for transductive transfer learning," *Seventh IEEE International Conference on Data Mining Workshops*, pp. 77–82, 2007.

[9] O. Kallenberg, *Foundations of Modern Probability*, Springer, 2002.

[10] P. Bartlett, M. Jordan, and J. McAuliffe, "Convexity, classification, and risk bounds," *J. Amer. Stat. Assoc.*, vol. 101, no. 473, pp. 138–156, 2006.

[11] G. Blanchard, G. Lee, and C. Scott, "Supplemental material," *NIPS* 2011.

[12] I. Steinwart and A. Christmann, *Support Vector Machines*, Springer, 2008.

[13] A. Gretton, K. Borgwardt, M. Rasch, B. Schölkopf, and A. Smola, "A kernel approach to comparing distributions," in *Proceedings of the 22nd AAAI Conference on Artificial Intelligence*, R. Holte and A. Howe, Eds., 2007, pp. 1637–1641.

[14] A. Gretton, K. Borgwardt, M. Rasch, B. Schölkopf, and A. Smola, "A kernel method for the two-sample-problem," in *Advances in Neural Information Processing Systems 19*, B. Schölkopf, J. Platt, and T. Hoffman, Eds., 2007, pp. 513–520.

[15] B. Sriperumbudur, A. Gretton, K. Fukumizu, B. Schölkopf, and G. Lanckriet, "Hilbert space embeddings and metrics on probability measures," *Journal of Machine Learning Research*, vol. 11, pp. 1517–1561, 2010.

[16] A. Christmann and I. Steinwart, "Universal kernels on non-standard input spaces," in *Advances in Neural Information Processing Systems 23*, J. Lafferty, C. K. I. Williams, J. Shawe-Taylor, R. Zemel, and A. Culotta, Eds., 2010, pp. 406–414.

[17] C. McDiarmid, "On the method of bounded differences," *Surveys in Combinatorics*, vol. 141, pp. 148–188, 1989.

[18] V. Koltchinskii, "Rademacher penalties and structural risk minimization," *IEEE Transactions on Information Theory*, vol. 47, no. 5, pp. 1902 – 1914, 2001.

[19] P. Bartlett and S. Mendelson, "Rademacher and Gaussian complexities: Risk bounds and structural results," *Journal of Machine Learning Research*, vol. 3, pp. 463–482, 2002.

[20] T. Joachims, "Making large-scale SVM learning practical," in *Advances in Kernel Methods - Support Vector Learning*, B. Schölkopf, C. Burges, and A. Smola, Eds., chapter 11, pp. 169–184. MIT Press, Cambridge, MA, 1999.

[21] J. Toedling, P. Rhein, R. Ratei, L. Karawajew, and R. Spang, "Automated in-silico detection of cell populations in flow cytometry readouts and its application to leukemia disease monitoring," *BMC Bioinformatics*, vol. 7, pp. 282, 2006.

[22] J. Wiens, *Machine Learning for Patient-Adaptive Ectopic Beat Classication*, Masters Thesis, Department of Electrical Engineering and Computer Science, Massachusetts Institute of Technology, 2010.

